# Active Support Vector Machine Classification

**O. L. Mangasarian**
Computer Sciences Dept.
University of Wisconsin
1210 West Dayton Street
Madison, WI 53706
*olvi@cs.wisc.edu*

**David R. Musicant**
Dept. of Mathematics and Computer Science
Carleton College
One North College Street
Northfield, MN 55057
*dmusican@carleton.edu*

## Abstract

An active set strategy is applied to the dual of a simple reformulation of the standard quadratic program of a linear support vector machine. This application generates a fast new dual algorithm that consists of solving a finite number of linear equations, with a typically large dimensionality equal to the number of points to be classified. However, by making novel use of the Sherman-Morrison-Woodbury formula, a much smaller matrix of the order of the original input space is inverted at each step. Thus, a problem with a 32-dimensional input space and 7 million points required inverting positive definite symmetric matrices of size $33\times33$ with a total running time of 96 minutes on a 400 MHz Pentium II. The algorithm requires no specialized quadratic or linear programming code, but merely a linear equation solver which is publicly available.

## 1 Introduction

Support vector machines (SVMs) [23, 5, 14, 12] are powerful tools for data classification. Classification is achieved by a linear or nonlinear separating surface in the input space of the dataset. In this work we propose a very fast simple algorithm, based on an active set strategy for solving quadratic programs with bounds [18]. The algorithm is capable of accurately solving problems with millions of points and requires nothing more complicated than a commonly available linear equation solver [17, 1, 6] for a typically small (100) dimensional input space of the problem.

Key to our approach are the following two changes to the standard linear SVM:

1. Maximize the margin (distance) between the parallel separating planes with respect to both orientation ($w$) as well as location relative to the origin ($\gamma$). See equation (7) below. Such an approach was also successfully utilized in the successive overrelaxation (SOR) approach of [15] as well as the smooth support vector machine (SSVM) approach of [12].

2. The error in the soft margin ($y$) is minimized using the 2-norm squared instead of the conventional 1-norm. See equation (7). Such an approach has also been used successfully in generating virtual support vectors [4].

These simple, but fundamental changes, lead to a considerably simpler positive definite dual problem with nonnegativity constraints only. See equation (8).

In Section 2 of the paper we begin with the standard SVM formulation and its dual and then give our formulation and its simpler dual. We corroborate with solid computational evidence that our simpler formulation does not compromise on generalization ability as evidenced by numerical tests in Section 4 on 6 public datasets. See Table 1. Section 3 gives our active support vector machine (ASVM) Algorithm 3.1 which consists of solving a system of linear equations in $m$ dual variables with a positive definite matrix. By invoking the Sherman-Morrison-Woodbury (SMW) formula (1) we need only invert an $(n+1) \times (n+1)$ matrix where $n$ is the dimensionality of the input space. This is a key feature of our approach that allows us to solve problems with millions of points by merely inverting much smaller matrices of the order of $n$. In concurrent work [8] Ferris and Munson also use the SMW formula but in conjunction with an interior point approach to solve massive problems based on our formulation (8) as well as the conventional formulation (6). Burges [3] has also used an active set method, but applied to the standard SVM formulation (2) instead of (7) as we do here. Both this work and Burges' appeal, in different ways, to the active set computational strategy of Moré and Toraldo [18]. We note that an *active set* computational strategy bears no relation to *active learning*. Section 4 describes our numerical results which indicate that the ASVM formulation has a tenfold testing correctness that is as good as the ordinary SVM, and has the capability of accurately solving massive problems with millions of points that cannot be attacked by standard methods for ordinary SVMs.

We now describe our notation and give some background material. All vectors will be column vectors unless transposed to a row vector by a prime $'$. For a vector $x \in R^n$, $x_+$ denotes the vector in $R^n$ with all of its negative components set to zero. The notation $A \in R^{m \times n}$ will signify a real $m \times n$ matrix. For such a matrix $A'$ will denote the transpose of $A$ and $A_i$ will denote the $i$-th row of $A$. A vector of ones or zeroes in a real space of arbitrary dimension will be denoted by $e$ or $0$, respectively. The identity matrix of arbitrary dimension will be denoted by $I$. For two vectors $x$ and $y$ in $R^n$, $x \perp y$ denotes orthogonality, that is $x'y = 0$. For $u \in R^m$, $Q \in R^{m \times m}$ and $B \subset \{1, 2, \ldots, m\}$, $u_B$ denotes $u_{i \in B}$, $Q_B$ denotes $Q_{i \in B}$ and $Q_{BB}$ denotes a principal submatrix of $Q$ with rows $i \in B$ and columns $j \in B$. The notation $\text{argmin}_{x \in S} \ f(x)$ denotes the set of minimizers in the set $S$ of the real-valued function $f$ defined on $S$. We use $:=$ to denote definition. The 2-norm of a matrix $Q$ will be denoted by $\|Q\|_2$. A separating plane, with respect to two given point sets $\mathcal{A}$ and $\mathcal{B}$ in $R^n$, is a plane that attempts to separate $R^n$ into two halfspaces such that each open halfspace contains points mostly of $\mathcal{A}$ or $\mathcal{B}$. A special case of the Sherman-Morrison-Woodbury (SMW) formula [9] will be utilized:

$$(I/\nu + HH')^{-1} = \nu(I - H(I/\nu + H'H)^{-1}H'), \tag{1}$$

where $\nu$ is a positive number and $H$ is an arbitrary $m \times k$ matrix. This formula enables us to invert a large $m \times m$ matrix by merely inverting a smaller $k \times k$ matrix.

## 2    The Linear Support Vector Machine

We consider the problem of classifying $m$ points in the $n$-dimensional real space $R^n$, represented by the $m \times n$ matrix $A$, according to membership of each point $A_i$ in the class $A+$ or $A-$ as specified by a given $m \times m$ diagonal matrix $D$ with $+1$'s or $-1$'s along its diagonal. For this problem the standard SVM with a linear kernel [23, 5] is given by the following quadratic program with parameter $\nu > 0$:

$$\min_{(w, \gamma, y) \in R^{n+1+m}} \nu e'y + \frac{1}{2}w'w \ \text{ s.t. } D(Aw - e\gamma) + y \geq e, \ y \geq 0. \tag{2}$$

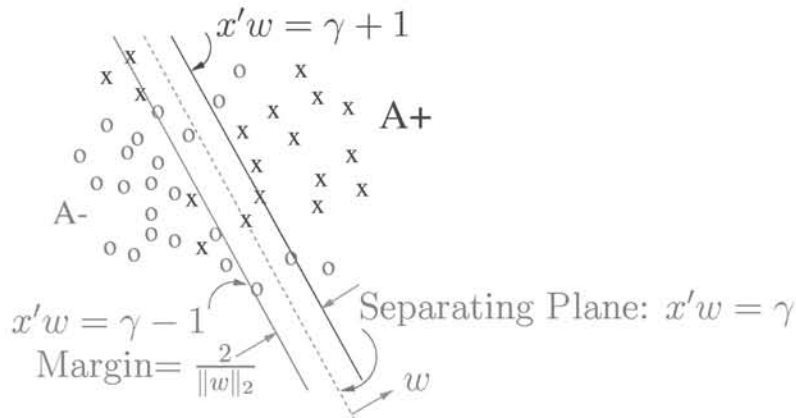

Figure 1: **The bounding planes (3) with a soft (i.e. with some errors) margin**
$2/\|w\|_2$, **and the plane (4) approximately separating** $A+$ **from** $A-$.

Here $w$ is the normal to the bounding planes:

$$x'w = \gamma \pm 1 \tag{3}$$

and $\gamma$ determines their location relative to the origin (Figure 1.) The plane $x'w = \gamma + 1$ bounds the $A+$ points, possibly with error, and the plane $x'w = \gamma - 1$ bounds the $A-$ points, also possibly with some error. The separating surface is the plane:

$$x'w = \gamma, \tag{4}$$

midway between the bounding planes (3). The quadratic term in (2), is twice the reciprocal of the square of the 2-norm distance $2/\|w\|_2$ between the two bounding planes of (3) (see Figure 1). This term maximizes this distance which is often called the "margin". If the classes are linearly inseparable, as depicted in Figure 1, then the two planes bound the two classes with a "soft margin". That is, they bound each set approximately with some error determined by the nonnegative error variable $y$:

$$\begin{array}{rclcll} A_iw & + & y_i & \geq & \gamma + 1, & \text{for } D_{ii} = 1, \\ A_iw & - & y_i & \leq & \gamma - 1, & \text{for } D_{ii} = -1. \end{array} \tag{5}$$

Traditionally the 1-norm of the error variable $y$ is minimized parametrically with weight $\nu$ in (2) resulting in an approximate separation as depicted in Figure 1. The dual to the standard quadratic linear SVM (2) [13, 22, 14, 7] is the following:

$$\min_{u \in R^m} \frac{1}{2} u' D A A' D u - e' u \ \text{ s.t. } e' D u = 0, \ \ 0 \leq u \leq \nu e. \tag{6}$$

The variables $(w, \gamma)$ of the primal problem which determine the separating surface (4) can be obtained from the solution of the dual problem above [15, Eqns. 5 and 7]. We note immediately that the matrix $DAA'D$ appearing in the dual objective function (6) is not positive definite in general because typically $m >> n$. Also, there is an equality constraint present, in addition to bound constraints, which for large problems necessitates special computational procedures such as SMO [21]. Furthermore, a one-dimensional optimization problem [15] must be solved in order to determine the locator $\gamma$ of the separating surface (4). In order to overcome all these difficulties as well as that of dealing with the necessity of having to essentially invert a very large matrix of the order of $m \times m$, we propose the following simple but critical modification of the standard SVM formulation (2). We change $\|y\|_1$ to $\|y\|_2^2$ which makes the constraint $y \geq 0$ redundant. We also append the term $\gamma^2$ to $w'w$. This in effect maximizes the margin between the parallel separating planes (3) with respect to both $w$ and $\gamma$ [15], that is with respect to both orientation and

location of the planes, rather that just with respect to $w$ which merely determines the orientation of the plane. This leads to the following reformulation of the SVM:

$$\min_{(w,\gamma,y) \in R^{n+1+m}} \nu \frac{y'y}{2} + \frac{1}{2}(w'w + \gamma^2) \text{ s.t. } D(Aw - e\gamma) + y \geq e. \quad (7)$$

the dual of this problem is [13]:

$$\min_{0 \leq u \in R^m} \frac{1}{2} u'(\frac{I}{\nu} + D(AA' + ee')D)u - e'u. \quad (8)$$

The variables $(w, \gamma)$ of the primal problem which determine the separating surface (4) are recovered directly from the solution of the dual (8) above by the relations:

$$w = A'Du, \ y = u/\nu, \ \gamma = -e'Du. \quad (9)$$

We immediately note that the matrix appearing in the dual objective function is positive definite and that there is no equality constraint and no upper bound on the dual variable $u$. The only constraint present is a simple nonnegativity one. These facts lead us to our simple finite active set algorithm which requires nothing more sophisticated than inverting an $(n + 1) \times (n + 1)$ matrix at each iteration in order to solve the dual problem (8).

## 3   ASVM (Active Support Vector Machine) Algorithm

The algorithm consists of determining a partition of the dual variable $u$ into nonbasic and basic variables. The nonbasic variables are those which are set to zero. The values of the basic variables are determined by finding the gradient of the objective function of (8) with respect to these variables, setting this gradient equal to zero, and solving the resulting linear equations for the basic variables. If any basic variable takes on a negative value after solving the linear equations, it is set to zero and becomes nonbasic. **This is the essence of the algorithm.** In order to make the algorithm converge and terminate, a few additional safeguards need to be put in place in order to allow us to invoke the Moré-Toraldo finite termination result [18]. The other key feature of the algorithm is a computational one and makes use of the SMW formula. This feature allows us to invert an $(n + 1) \times (n + 1)$ matrix at each step instead of a much bigger matrix of order $m \times m$.

Before stating our algorithm we define two matrices to simplify notation as follows:

$$H = D[A \ \ -e], \quad Q = I/\nu + HH'. \quad (10)$$

With these definitions the dual problem (8) becomes

$$\min_{0 \leq u \in R^m} f(u) := \frac{1}{2} u'Qu - eu. \quad (11)$$

It will be understood that within the ASVM Algorithm, $Q^{-1}$ will always be evaluated using the SMW formula and hence only an $(n+1) \times (n+1)$ matrix is inverted. We state our algorithm now. Note that commented (%) parts of the algorithm are not needed in general and were rarely used in our numerical results presented in Section 4. The essence of the algorithm is displayed in the two boxes below.

**Algorithm 3.1 Active SVM (ASVM) Algorithm for (8).**

| | |
|---|---|
| (0) | Start with $u^0 := (Q^{-1}e)_+$. For $i = 1, 2, \ldots,$ having $u^i$ compute $u^{i+1}$ as follows. |
| (1) | Define $B^i := \{j \mid u^i_j > 0\}$, $N^i := \{j \mid u^i_j = 0\}$. |
| (2) | Determine $$u^{i+1}_{B^i} := (Q^{-1}_{B^iB^i}e_{B^i})_+, \ u^{i+1}_{N^i} := 0.$$ Stop if $u^{i+1}$ is the global solution, that is if $0 \leq u^{i+1} \perp Qu^{i+1} - e \geq 0$. |

(2a)  % If $f(u^{i+1}) \geq f(u^i)$, then go to (4a).

(2b)  % If $0 \leq u^{i+1}_{B^{i+1}} \perp Q_{B^{i+1}B^{i+1}} u^{i+1}_{B^{i+1}} - e_{B^{i+1}} \geq 0$, then $u^{i+1}$ is a global solution on the face of active constraints: $u_{N^i} = 0$. Set $u^i := u^{i+1}$ and go to (4b).

(3)  $\boxed{\text{Set } i := i+1 \text{ and go to (1).}}$

(4a)  % Move in the direction of the global minimum on the face of active constraints, $u_{N^i} = 0$. Set $\tilde{u}^{i+1}_{B^i} := Q^{-1}_{B^i B^i} e_{B^i}$ and $u^{i+1}_{B^i} := argmin_{0 \leq \lambda \leq 1}\{f(u^i_{B^i} + \lambda(\tilde{u}^{i+1}_{B^i} - u^i_{B^i})) \mid u^i_{B^i} + \lambda(\tilde{u}^{i+1}_{B^i} - u^i_{B^i}) \geq 0\}$. If $u^{i+1}_j = 0$ for some $j \in B^i$, set $i := i+1$ and go to (1). Otherwise $u^{i+1}$ is a global minimum on the face $u_{N^i} = 0$, and go to (4b).

(4b)  % Iterate a gradient projection step. Set $k := 0$ and $\tilde{u}^k := u^i$. Iterate $\tilde{u}^{k+1} := argmin_{0 \leq \lambda \leq 1} f(\tilde{u}^k - \lambda(\tilde{u}^k - (Q\tilde{u}^k - e))_+)$, $k := k+1$ until $f(\tilde{u}^k) < f(u^i)$. Set $u^{i+1} := \tilde{u}^k$. Set $i := i+1$ and go to (1).

**Remark 3.2** *All commented (%) parts of the algorithm are optional and are not usually implemented unless the algorithm gets stuck, which it rarely did on our examples. Hence our algorithm is particularly simple and consists of steps (0), (1),(2) and (3). The commented parts were inserted in order to comply with the active set strategy of Moré-Toraldo result [18] for which they give finite termination.*

**Remark 3.3** *The iteration in step (4b) is a gradient projection step which is guaranteed to converge to the global solution of (8) [2, pp 223-225] and is placed here to ensure that the strict inequality $f(\tilde{u}^k) < f(u^i)$ eventually holds as required in [18]. Similarly, the step in (4a) ensures that the function value does not increase when it remains on the same face, in compliance with [18, Algortihm BCQP(b)].*

## 4  Numerical Implementation and Comparisons

We implemented ASVM in Visual C++ 6.0 under Windows NT 4.0. The experiments were run on the UW-Madison Data Mining Institute Locop2 machine, which utilizes a 400 MHz Pentium II Xeon Processor and a maximum of 2 Gigabytes of memory available per process. We wrote all the code ourselves except for the linear equation solver, for which we used CLAPACK [1, 6]. Our stopping criterion for ASVM is triggered when the error bound residual [16] $\|u - (u - Qu + e)_+\|$, which is zero at the solution of (11), goes below 0.1.

The first set of experiments are designed to show that our reformulation (8) of the SVM (7) and its associated algorithm ASVM yield similar performance to the standard SVM (2), referred to here as SVM-QP. For six datasets available from the UCI Machine Learning Repository [19], we performed tenfold cross validation in order to compare test set accuracies between ASVM and SVM-QP. We implemented SVM-QP using the high-performing CPLEX barrier quadratic programming solver [10], and utilized a tuning set for both algorithms to find the optimal value of the parameter $\nu$, using the default stopping criterion of CPLEX. Altering the CPLEX default stopping criterion to match that of ASVM did not result in significant change in timing relative to ASVM, but did reduce test set correctness.

In order to obtain additional timing comparison information, we also ran the well-known SVM optimized algorithm SVM$^{light}$ [11]. Joachims, the author of SVM$^{light}$, provided us with the newest version of the software (Version 3.10b) and advice on setting the parameters. All features for these experiments were normalized to the range $[-1, +1]$ as recommended in the SVM$^{light}$ documentation. We chose to use

| Dataset m x n | Algorithm | Training Correctness | Testing Correctness | Time (CPU sec) | Dataset m x n | Algorithm | Training Correctness | Testing Correctness | Time (CPU sec) |
|---|---|---|---|---|---|---|---|---|---|
| Liver Disorders | CPLEX | 70.76% | 68.41% | 7.87 | Ionosphere | CPLEX | 92.81% | 88.60% | 9.84 |
| | SVM*light* | 70.37% | 68.12% | 0.26 | | SVM*light* | 92.81% | 88.60% | 0.23 |
| 345 x 6 | ASVM | 70.40% | 67.25% | 0.03 | 351 x 34 | ASVM | 93.29% | 87.75% | 0.26 |
| Cleveland Heart | CPLEX | 87.50% | 84.20% | 4.17 | Tic Tac Toe | CPLEX | 65.34% | 65.34% | 206.52 |
| | SVM*light* | 87.50% | 84.20% | 0.17 | | SVM*light* | 65.34% | 65.34% | 0.23 |
| 297 x 13 | ASVM | 87.24% | 85.56% | 0.05 | 958 x 9 | ASVM | 70.27% | 69.72% | 0.05 |
| Pima Diabetes | CPLEX | 77.36% | 76.95% | 128.90 | Votes | CPLEX | 96.02% | 95.85% | 27.26 |
| | SVM*light* | 77.36% | 76.95% | 0.19 | | SVM*light* | 96.02% | 95.85% | 0.06 |
| 768 x 8 | ASVM | 78.04% | 78.12% | 0.08 | 435 x 16 | ASVM | 96.73% | 96.07% | 0.09 |

Table 1: **ASVM compared with conventional SVM-QP (CPLEX and SVM$^{light}$) on UCI datasets. ASVM test correctness is comparable to SVM-QP, with timing much faster than CPLEX and faster than or comparable to SVM$^{light}$.**

| # of Points | Iterations | Training Correctness | Testing Correctness | Time (CPU min) |
|---|---|---|---|---|
| 4 million | 5 | 86.09% | 86.06% | 38.04 |
| 7 million | 5 | 86.10% | 86.28% | 95.57 |

Table 2: **Performance of ASVM on NDC generated datasets in $R^{32}$. ($\nu = 0.01$)**

the default termination error criterion in SVM$^{light}$ of 0.001, which is actually a less stringent criterion than the one we used for ASVM. This is because the criterion we used for ASVM (see above) is an aggregate over the errors for all points, whereas the SVM$^{light}$ criterion reflects a minimum error threshold for each point.

The second set of experiments show that ASVM performs well on massive datasets. We created synthetic data of Gaussian distribution by using our own NDC Data Generator [20] as suggested by Usama Fayyad. The results of our experiments are shown in Table 2. We did try to run SVM$^{light}$ on these datasets as well, but we ran into memory difficulties. Note that for these experiments, all the data was brought into memory. As such, the running time reported consists of the time used to actually solve the problem to termination excluding I/O time. This is consistent with the measurement techniques used by other popular approaches [11, 21]. Putting all the data in memory is simpler to code and results in faster running times. However, it is not a fundamental requirement of our algorithm — block matrix multiplications, incremental evaluations of $Q^{-1}$ using another application of the SMW formula, and indices on the dataset can be used to create an efficient disk based version of ASVM.

# 5    Conclusion

A very fast, finite and simple algorithm, ASVM, capable of classifying massive datasets has been proposed and implemented. ASVM requires nothing more complex than a commonly available linear equation solver for solving small systems with few variables even for massive datasets. Future work includes extensions to parallel processing of the data, handling very large datasets directly from disk as well as extending our approach to nonlinear kernels.

# Acknowledgements

We are indebted to our colleagues Thorsten Joachims for helping us to get SVM$^{light}$ running significantly faster on the UCI datasets, and to Glenn Fung for his efforts in running the experiments for revisions of this work. Research described in this Data Mining Institute Report 00-04, April 2000, was supported by National Science Foundation Grants CCR-9729842 and CDA-9623632, by Air Force Office of Scientific Research Grant F49620-00-1-0085 and by Microsoft.

# References

[1] E. Anderson, Z. Bai, C. Bischof, J. Demmel, J. Dongarra, J. Du Cros, A. Greenbaum, S. Hammarling, A. McKenney, S. Ostrouchov, and D. Sorensen. *LA-PACK User's Guide*. SIAM, Philadelphia, Pennsylvania, second edition, 1995.

[2] D. P. Bertsekas. *Nonlinear Programming*. Athena Scientific, Belmont, MA, second edition, 1999.

[3] C. J. C. Burges. A tutorial on support vector machines for pattern recognition. *Data Mining and Knowledge Discovery*, 2(2):121–167, 1998.

[4] C. J. C. Burges and B. Schölkopf. Improving the accuracy and speed of support vector machines. In M. C. Mozer, M. I. Jordan, and T. Petsche, editors, *Advances in Neural Information Processing Systems -9-*, pages 375–381, Cambridge, MA, 1997. MIT Press.

[5] V. Cherkassky and F. Mulier. *Learning from Data - Concepts, Theory and Methods*. John Wiley & Sons, New York, 1998.

[6] CLAPACK. f2c'ed version of LAPACK. http://www.netlib.org/clapack.

[7] N. Cristianini and J. Shawe-Taylor. *An Introduction to Support Vector Machines*. Cambridge University Press, Cambridge, 2000.

[8] M. C. Ferris and T. S. Munson. Interior point methods for massive support vector machines. Technical Report 00-05, Computer Sciences Department, University of Wisconsin, Madison, Wisconsin, May 2000.

[9] G. H. Golub and C. F. Van Loan. *Matrix Computations*. The John Hopkins University Press, Baltimore, Maryland, 3rd edition, 1996.

[10] ILOG, Incline Village, Nevada. *CPLEX 6.5 Reference Manual*, 1999.

[11] T. Joachims. $\text{SVM}^{light}$, 1998. http://www-ai.informatik.uni-dortmund.de/FORSCHUNG/VERFAHREN/SVM_LIGHT/svm_light.eng.html.

[12] Yuh-Jye Lee and O. L. Mangasarian. SSVM: A smooth support vector machine. *Computational Optimization and Applications*, 2000.

[13] O. L. Mangasarian. *Nonlinear Programming*. SIAM, Philadelphia, PA, 1994.

[14] O. L. Mangasarian. Generalized support vector machines. In A. Smola, P. Bartlett, B. Schölkopf, and D. Schuurmans, editors, *Advances in Large Margin Classifiers*, pages 135–146, Cambridge, MA, 2000. MIT Press.

[15] O. L. Mangasarian and D. R. Musicant. Successive overrelaxation for support vector machines. *IEEE Transactions on Neural Networks*, 10:1032–1037, 1999.

[16] O. L. Mangasarian and J. Ren. New improved error bounds for the linear complementarity problem. *Mathematical Programming*, 66:241–255, 1994.

[17] MATLAB. *User's Guide*. The MathWorks, Inc., Natick, MA 01760, 1992.

[18] J. J. Moré and G. Toraldo. Algorithms for bound constrained quadratic programs. *Numerische Mathematik*, 55:377–400, 1989.

[19] P. M. Murphy and D. W. Aha. UCI repository of machine learning databases, 1992. www.ics.uci.edu/~mlearn/MLRepository.html.

[20] D. R. Musicant. NDC: normally distributed clustered datasets, 1998. www.cs.wisc.edu/~musicant/data/ndc/.

[21] J. Platt. Sequential minimal optimization: A fast algorithm for training support vector machines. In Schölkopf et al. [22], pages 185–208.

[22] B. Schölkopf, C. Burges, and A. Smola (editors). *Advances in Kernel Methods: Support Vector Machines*. MIT Press, Cambridge, MA, 1998.

[23] V. N. Vapnik. *The Nature of Statistical Learning Theory*. Springer, NY, 1995.
